# A configurable analog VLSI neural network with spiking neurons and self-regulating plastic synapses which classifies overlapping patterns

**M. Giulioni**[*]
Italian National Inst. of Health, Rome, Italy
INFN-RM2, Rome, Italy
giulioni@roma2.infn.it

**M. Pannunzi**
Italian National Inst. of Health, Rome, Italy
INFN-RM1, Rome, Italy

**D. Badoni**
INFN-RM2, Rome, Italy

**V. Dante**
Italian National Inst. of Health, Rome, Italy
INFN-RM1, Rome, Italy

**P. Del Giudice**
Italian National Inst. of Health, Rome, Italy
INFN-RM1, Rome, Italy

## Abstract

We summarize the implementation of an analog VLSI chip hosting a network of 32 integrate-and-fire (IF) neurons with spike-frequency adaptation and 2,048 Hebbian plastic bistable spike-driven stochastic synapses endowed with a self-regulating mechanism which stops unnecessary synaptic changes. The synaptic matrix can be flexibly configured and provides both recurrent and AER-based connectivity with external, AER compliant devices. We demonstrate the ability of the network to efficiently classify overlapping patterns, thanks to the self-regulating mechanism.

## 1 Introduction

Neuromorphic analog, VLSI devices [12] try to derive organizational and computational principles from biologically plausible models of neural systems, aiming at providing in the long run an electronic substrate for innovative, bio-inspired computational paradigms.

In line with standard assumptions in computational neuroscience, neuromorphic devices are endowed with adaptive capabilities through various forms of plasticity in the synapses which connect the neural elements. A widely adopted framework goes under the name of Hebbian learning, by which the efficacy of a synapse is potentiated (the post-synaptic effect of a spike is enhanced) if the pre- and post-synaptic neurons are simultaneously active on a suitable time scale. Different mechanisms have been proposed, some relying on the average firing rates of the pre- and post-synaptic neurons, (rate-based Hebbian learning), others based on tight constraints on the time lags between pre- and post-synaptic spikes ("Spike-Timing-Dependent-Plasticity").

The synaptic circuits described in what follows implement a stochastic version of rate-based Hebbian learning. In the last decade, it has been realized that general constraints plausibly met by any concrete implementation of a synaptic device in a neural network, bear profound consequences on

---

[*]http://neural.iss.infn.it/

the capacity of the network as a memory system. Specifically, once one accepts that a synaptic element can neither have an unlimited dynamic range (i.e. synaptic efficacy is bounded), nor can it undergo arbitrarily small changes (i.e. synaptic efficacy has a finite analog depth), it has been proven ([1], [7]) that a deterministic learning prescription implies an extremely low memory capacity, and a severe "palimpsest" property: new memories quickly erase the trace of older ones. It turns out that a stochastic mechanism provides a general, logically appealing and very efficient solution: given the pre- and post-synaptic neural activities, the synapse is still made eligible for changing its efficacy according to a Hebbian prescription, but it actually changes its state with a given probability. The stochastic element of the learning dynamics would imply ad hoc new elements, were it not for the fact that for a spike-driven implementation of the synapse, the noisy activity of the neurons in the network can provide the needed "noise generator" [7]. Therefore, for an efficient learning electronic network, the implementation of the neuron as a spiking element is not only a requirement of "biological plausibility", but a compelling computational requirement. Learning in networks of spiking IF neurons with stochastic plastic synapses has been studied theoretically [7], [10], [2], and stochastic, bi-stable synaptic models have been implemented in silicon [8], [6]. One of the limitations so far, both at the theoretical and the implementation level, has been the artificially simple statistics of the stimuli to be learnt (e.g., no overlap between their neural representations). Very recently in [4] a modification of the above stochastic, bi-stable synaptic model has been proposed, endowed with a regulatory mechanism termed "stop learning" such that synaptic up or down-regulation depends on the average activity of the postsynaptic neuron in the recent past; a synapse pointing to a neuron that is found to be highly active, or poorly active, should not be further potentiated or depressed, respectively. The reason behind the prescription is essentially that for correlated patterns to be learnt by the network, a successful strategy should de-emphasize the coherent synaptic Hebbian potentiation that would result for the overlapping part of the synaptic matrix, and that would ultimately spoil the ability to distinguish the patterns. A detailed learning strategy along this line was proven in [13] to be appropriate for linearly separable patterns for a Perceptron-like network; the extension to spiking and recurrent networks is currently studied.

In section 2 we give an overview of the chip architecture and of the implemented synaptic model. In section 3 we show an example of the measures effectuated on the chip useful to characterize the synaptic and neuronal parameters. In section 4 we report some characterization results compared with a theoretical prediction obtained from a chip-oriented simulation. The last paragraph describes chip performances in a simple classification task, and illustrate the improvement brought about by the stop-learning mechanism.

## 2    Chip architecture and main features

The chip, already described in [3] implements a recurrent network of 32 integrate-and-fire neurons with spike-frequency adaptation and bi-stable, stochastic, Hebbian synapses. A completely reconfigurable synaptic matrix supports up to all-to-all recurrent connectivity, and AER-based external connectivity. Besides establishing an arbitrary synaptic connectivity, the excitatory/inhibitory nature of each synapse can also be set.

The implemented neuron is the IF neuron with constant leakage term and a lower bound for the membrane potential $V(t)$ introduced in [12] and studied theoretically in [9]. The circuit is borrowed from the low-power design described in [11], to which we refer the reader for details. Only 2 neurons can be directly probed (i.e., their "membrane potential" sampled), while for all of them the emitted spikes can be monitored via AER [5]. The dendritic tree of each neuron is composed of up to 31 activated recurrent synapses and up to 32 activated external, AER ones. For the recurrent synapses, each impinging spike triggers short-time (and possibly long-term) changes in the state of the synapse, as detailed below. Spikes from neurons outside the chip come in the form of AER events, and are targeted to the correct AER synapse by the X-Y Decoder. Synapses which are set to be excitatory, either AER or recurrent are plastic; inhibitory synapses are fixed. Spikes generated by the neurons in the chip are arbitrated for access to the AER bus for monitoring and/or mapping to external targets.

The synaptic circuit described in [3] implements the model proposed in [4] and briefly motivated in the Introduction. The synapse possesses only two states of efficacy (a bi-stable device): the internal synaptic dynamics is associated with an internal variable $X$; when $X > \theta_X$ the efficacy is set to be

potentiated, otherwise is set to be depressed. $X$ is subjected to short-term, spike-driven dynamics: upon the arrival of an impinging spike, $X$ is candidate for an upward or downward jump, depending on the instantaneous value of the post-synaptic potential $V_{post}$ being above or below a threshold $\theta_V$. The jump is actually performed or not depending on a further variable as explained below. In the absence of intervening spikes $X$ is forced to drift towards a "high" or "low" value depending on whether the last jump left it above or below $\theta_X$. This preserves the synaptic efficacy on long time scale.

A further variable is associated with the post-synaptic neuron dynamics, which essentially measures the average firing activity. Following [4], by analogy with the role played by the intracellular concentration of calcium ions upon spike emission, we will call it a "calcium variable" $C(t)$. $C(t)$ undergoes an upward jump when the postsynaptic neuron emits a spike, and linearly decays between two spikes. It therefore integrates the spikes sequence and, when compared to suitable thresholds as detailed below, it determines which candidate synaptic jumps will be allowed to occur; for example, it can constrain the synapse to stop up-regulating because the post-synaptic neuron is already very active. $C(t)$ acts as a regulatory element of the synaptic dynamics.

The resulting short-term dynamics for the internal synaptic variable $X$ is described by the following conditions: $X(t) \rightarrow X(t) + J_{up}$ if $V_{post}(t) > \theta_V$ and $V_{TH1} < C(t) < V_{TH3}$; $X(t) \rightarrow X(t) - J_{dw}$ if $V_{post}(t) \leq \theta_V$ and $V_{TH1} < C(t) < V_{TH2}$ where $J_{up}$ and $J_{dw}$ are positive constants. Detailed description of circuits implementing these conditions can be found in [3].

In figure 1 we illustrate the effect of the calcium dynamics on $X$. Increasing input forces the post-synaptic neuron to fire at increasing frequencies. As long as $C(t) < V_{TH2} = V_{TH3}$ $X$ undergoes both up and down jumps. When $C(t) > V_{TH2} = V_{TH3}$ jumps are inhibited and $X$ is forced to drift towards its lower bound.

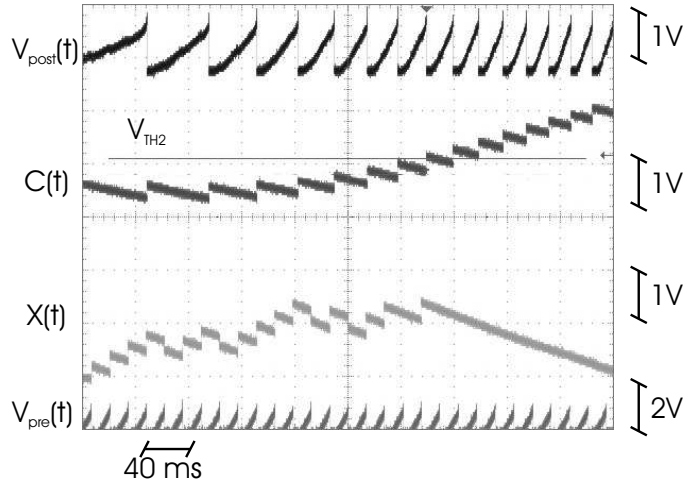

Figure 1: Illustrative example of the stop-learning mechanism (see text). Top to bottom: post-synaptic neuron potential $V_{post}$, calcium variable $C$, internal synaptic variable $X$, pre-synaptic neuron potential $V_{pre}$

## 3  LTP/LTD probabilities: measurement VS chip-oriented simulation

We report synapse potentiation (LTP) / depression (LTD)from the chip and we compare experimental results to simulations.

For each synapse in a subset of 31, we generate a pre-synaptic poisson spike train at 70 Hz. The post synaptic neuron is forced to fire a poisson spike train by applying an external DC current and a poisson train of inhibitory spikes through AER. Setting to zero both the potentiated and depressed efficacies, the activity of the post-synaptic neuron can be easily tuned by varying the amplitude of the DC current and the frequency of the inhibitory AER train. We initialize the 31 (AER) synapses to depressed (potentiated) and we monitor the post-synaptic neuron activity during a stimulation

trial lasting 0.5 seconds. At the end of the trial we read the synaptic state using an AER protocol developed to this purpose. For each chosen value of the post-synaptic firing rate, we evaluate the probability to find synapses in a potentiated (depressed) state repeating the test 50 times. The results reported in figure 2 (solid lines) represent the average LTP and LTD probabilities per trail over the 31 synapses. Tests were performed both with active and inactive Calcium mechanism. When calcium mechanism is inactive, the LTP is monotonically increasing with the post-synaptic firing rate while when the calcium circuit is activated the LTP probability has a max form $V_{post}$ around 80 Hz.

Identical tests were also run in simulation (dashed curves in figure 2). For the purpose of a meaningful comparison with the chip behaviour relevant parameter affecting neural and synaptic dynamics and their distributions (due to inhomogenities and mismatches) are characterized.

Simulated and measured data are in qualitative agreement. The parameters we chose for these tests are the same used for the classification task described in the next paragraph.

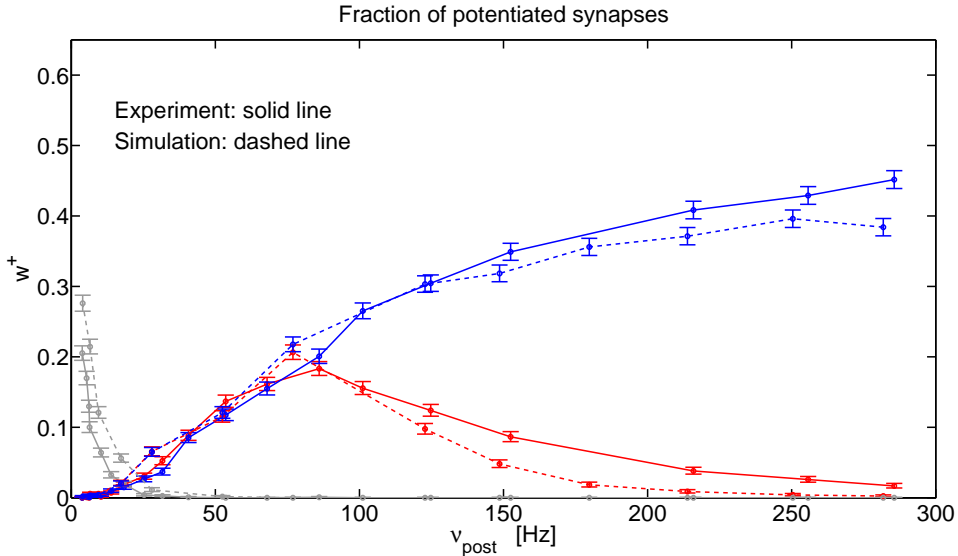

Figure 2: Transition probabilities. Red and blue lines are LTP probabilities with and without calcium stop-learning mechanism respectively. Gray lines are LTD probabilities without calcium stop-learning mechanism, the case LTD with Ca mechanism is not shown. Error bars are standard deviations over the 50 trials

## 4   Learning overlapping patterns

We configured the synaptic matrix to have a perceptron like network with 1 output and 32 inputs (32 AER synapses). 31 synapses are set as plastic excitatory ones, the 32nd is set as inhibitory and used to modulate the post-synaptic neuron activity. Our aim is to teach the perceptron to classify two patterns through a semi-supervised learning strategy: "Up" and "Down". We expect that after learning the perceptron will respond with high output frequency for pattern "Up" and with low output frequency for pattern "Down". The self regulating Ca mechanism is exploited to improve performances when Up and Down patterns have a significant overlap. The learning is semi-supervised: for each pattern a "teacher" input is sent to the output neuron steering its activity to be high or low, as desired. At the end of the learning period the "teacher" is turned off and the perceptron output is driven only by the input stimuli: in this conditions its classification ability is tested.

We present learning performances for input patterns with increasing overlap, and demonstrate the effect of the stop learning mechanism (overlap ranging from 6 to 14 synapses).

Upon stimulation active pre-synaptic inputs are poisson spike trains at 70 Hz, while inactive inputs are poisson spike trains at 10 Hz. Each trial lasts half a second. Up and Down patterns are randomly presented with equal probability. The teaching signal, a combination of an excitatory constant cur-

rent and of an inhibitory AER spike train, forces the output firing rate to 50 or 0.5 Hz. One run lasts for 150 trials which is sufficient for the stabilization of the output frequencies. At the end of each trial we turn off the teaching signal, we freeze the synaptic dynamics and we read the state of each synapse using an AER protocol developed for this purpose. In these conditions we performed a 5 seconds test ("Checking Phase") to measure the perceptron frequencies when pattern Up or pattern Down are presented. Each experiment includes 50 runs. For each run we change: a) the "definition" of patterns Up and Down: inputs activated by pattern Up and Down are chosen randomly at the beginning of each run; b) the initial synaptic state, with the constraint that only about 30 % of the synapses are potentiated; c) the stimulation sequence.

For the first experiment we turned off the stop learning mechanism and we chose orthogonal patterns. In this case the perceptron was able to correctly classify the stimuli: after about 50 trials, choosing a suitable threshold, one can discriminate the perceptron ouput to different patterns (lower left panel on figure 4). The output frequency separation slightly increases until trial number 100 remaining almost stable after that point.

We then studied the case of overlapped patterns both with active and inactive Calcium mechanism. We repeated the experiment with an increasing overlap: 6, 10 and 14. (implying an increase in the coding level from 0.5 for the orthogonal case to 0.7 for the overlap equal to 14). Only the threshold $K_{high}^{up}$ is active (the threshold above which up jumps are inhibnited). The Calcium circuit parameters are tuned so that the Ca variable passes $K_{high}^{up}$ for the mean firing rate of the post-synaptic neuron around 80 Hz. We show in figure 3 the distributions of the potentiated fraction of the synapses over the 50 runs at different stages along the run for overlap 10 with inactive (upper panels) and active (lower panels) calcium mechanism. We divided synapses in three subgroups: Up (red) synapses with pre-synaptic input activated solely by Up pattern, Down (blue) synapses with pre-synaptic inputs activated only by Down pattern, and Overlap (green) synapses with pre-synpatic inputs activated by both pattern Up and Down. The state of the synapses is recorded after every learning step. Accumulating statistics over the 50 runs we obtain the distributions reported in figure 3. The fraction of potentiated synapses is calculated over the number of synapses belonging to each subgroup. When the stop learning mechanism is inactive, at the end of the experiment, the green

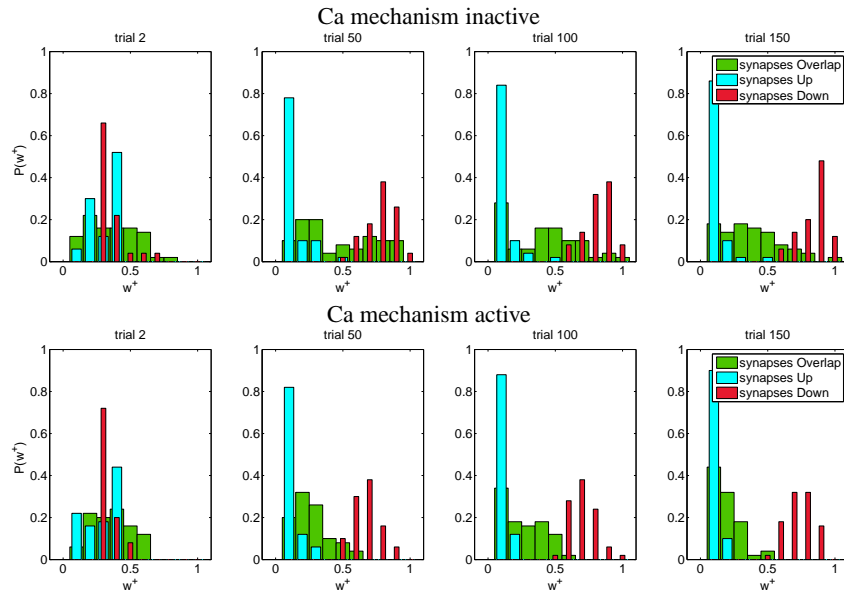

Figure 3: Distribution of the fraction of potentiated synapses. The number of inputs belonging to both patterns is 10.

distribution of overlap synapses is broad, when the Calcium mechanism is active, synapses overlap tend to be depotentiated. This result is the "microscopic" effect of the stop learning mechanism: once the number of potentiated synapses is sufficient to drive the perceptron output frequency above 80 Hz, the overlap synapses tend to be depotentiated. Overlap synapses would be pushed half of the

times to the potentiated state and half of the times to the depressed state, so that it is more likely for the Up synapses to reach earlier the potentiated state. When the stop learning mechanism is active, the potentiated synapses are enogh to drive the output neuron about 80 Hz, further potentiation is inhibited for all synapses so that overlap synapses get depressed on average. This happens under the condition that the transition probability are sufficiently small to avoid that at each trial the learning is completely disrupted. The distribution of the output frequencies for increasing overlap is illustrated in figure 4 (Ca mechanism inactive in the upper panels, active for the lower panels). The frequencies are recorded during the "checking phase". In blue the histograms of the output frequency for the down pattern, in red those for up pattern. It is clear from the figure that the output frequency distribution remain well separated even for high overlap when the Calcium mechanism is active.

A quantitative parameter to describe the distribution separation is

$$\delta = \frac{\overline{\nu}_{up} - \overline{\nu}_{dn}}{\sigma^2_{\nu_{up}} + \sigma^2_{\nu_{dn}}} \tag{1}$$

$\delta$ values are summarized in table 1.

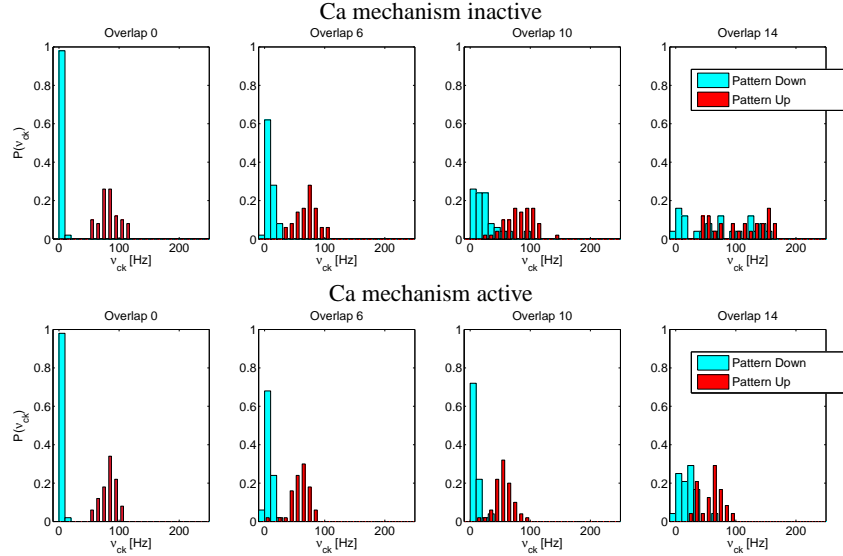

Figure 4: Distributions of perceptron frequencies after learning two overlapped patterns. Blue bars refer to pattern Down stimulation, red bars refers to pattern Up. Each panel refers to overlap.

Table 1: Discrimination power [seconds]

|        | overlap 0 | overlap 6 | overlap 10 | overlap 14 |
|--------|-----------|-----------|------------|------------|
| Ca OFF | 4.39      | 1.87      | 1.59       | 0.99       |
| Ca ON  | 5.29      | 2.20      | 1.88       | 1.66       |

For each run the number of potentiated synapses is different due to the random choices of Up, Down and Overlap synapses for each run and the mismatches affecting the behavior of different synapses. The failure of the discrimination for high overlap in the absence of this stop learning mechanism is due to the fact that the number of potentiated synapses can overcome the effect of the teaching signal for the down pattern. The Calcium mechanism, defining a maximum number of allowed potentiated synapses, limits this problem. This offer the possibility of establishing a priori threshold to discriminate the perceptron outputs on the basis of the frequency corresponding to the maximum value of the LTP probability curve.

# 5 Conclusions

We briefly illustrate an analog VLSI chip implementing a network of 32 IF neurons and 2,048 reconfigurable, Hebbian, plastic, stop-learning synapses. Circuit parameters has been measured as well as their dispersion across the chip. Using these data a chip-oriented simulation was set up and its results, compared to experimental ones, demonstrate that circuits behavior follow the theoretical predictions. Once configured the network as a perceptron (31 AER synapses and one output neuron), a classification task has been performed. Stimuli with an increasing overlap have been used. The results show the ability of the network to efficiently classify the presented patterns as well as the improvement of the performances due to the calcium stop-learning mechanism.

# References

[1] D.J. Amit and S. Fusi. *Neural Computation*, 6:957, 1994.

[2] D.J. Amit and G. Mongillo. *Neural Computation*, 15:565, 2003.

[3] D. Badoni, M. Giulioni, V. Dante, and P. Del Giudice. In *Proc. IEEE International Symposium on Circuits and Systems ISCAS06*, pages 1227–1230, 2006.

[4] J.M. Brader, W. Senn, and S. Fusi. *Neural Computation (in press)*, 2007.

[5] V. Dante, P. Del Giudice, and A. M. Whatley. The neuromorphic engineer newsletter. 2005.

[6] E. Chicca et al. *IEEE Transactions on Neural Networks*, 14(5):1297, 2003.

[7] S. Fusi. *Biological Cybernetics*, 87:459, 2002.

[8] S. Fusi, M. Annunziato, D. Badoni, A. Salamon, and D.J. Amit. *Neural Computation*, 12:2227, 2000.

[9] S. Fusi and M. Mattia. *Neural Computation*, 11:633, 1999.

[10] P. Del Giudice, S. Fusi, and M. Mattia. *Journal of Physiology Paris*, 97:659, 2003.

[11] G. Indiveri. In *Proc. IEEE International Symposium on Circuits and Systems*, 2003.

[12] C. Mead. *Analog VLSI and neural systems*. Addison-Wesley, 1989.

[13] W. Senn and S. Fusi. *Neural Computation*, 17:2106, 2005.

